# Worst-case bounds on the quality of max-product fixed-points

**Meritxell Vinyals**
Artificial Intelligence Research Institute (IIIA)
Spanish Scientific Research Council (CSIC)
Campus UAB, Bellaterra, Spain
`meritxell@iiia.csic.es`

**Jesús Cerquides**
Artificial Intelligence Research Institute (IIIA)
Spanish Scientific Research Council (CSIC)
Campus UAB, Bellaterra, Spain
`cerquide@iiia.csic.es`

**Alessandro Farinelli**
Department of Computer Science
University of Verona
Strada le Grazie, 15,Verona, Italy
`alessandro.farinelli@univr.it`

**Juan Antonio Rodríguez-Aguilar**
Artificial Intelligence Research Institute (IIIA)
Spanish Scientific Research Council (CSIC)
Campus UAB, Bellaterra, Spain
`jar@iiia.csic.es`

## Abstract

We study worst-case bounds on the quality of any fixed point assignment of the max-product algorithm for Markov Random Fields (MRF). We start providing a bound independent of the MRF structure and parameters. Afterwards, we show how this bound can be improved for MRFs with specific structures such as bipartite graphs or grids. Our results provide interesting insight into the behavior of max-product. For example, we prove that max-product provides very good results (at least $90\%$ optimal) on MRFs with large variable-disjoint cycles[1].

## 1   Introduction

Graphical models such as Markov Random Fields (MRFs) have been successfully applied to a wide variety of applications such as image understanding [1], error correcting codes [2], protein folding [3] and multi-agent systems coordination [4]. Many of these practical problems can be formulated as finding the maximum a posteriori (MAP) assignment, namely the most likely joint variable assignment in an MRF. The MAP problem is NP-hard [5], thus requiring approximate methods.

Here we focus on a particular MAP approximate method: the (loopy) max-product belief propagation [6, 7]. Max-product's popularity stems from its very good empirical performance on general MRFs [8, 9, 10, 11], but it comes with few theoretical guarantees. Concretely, max-product is known to be correct in acyclic and single-cycle MRFs [11], although convergence is only guaranteed in the acyclic case. Recently, some works have established that max-product is guarantee to return the optimal solution, if it converges, on MRFs corresponding to some specific problems, namely: (i) weighted b-matching problems [12, 13]; (ii) maximum weight independent set problems [14]; or (iii) problems whose equivalent nand Markov random field (NMRF) is a perfect graph [**?**]. For weighted b-matching problems with a bipartite structure Huang and Jebara [15] establish that max-product algorithm always converges to the optimal.

Despite these guarantees provided in these particular cases, for arbitrary MRFs little is known on the quality of the max-product fixed-point assignments. To the best of our knowledge, the only result in this line is the work of Wainwright et al. [16] where, given any arbitrary MRF, authors derive an upper bound on the absolute error of the max-product fixed-point assignment. This bound

is calculated after running the max-sum algorithm and depends on the particular MRF (structure and parameters) and therefore provide no guarantees on the quality of max-product assignments on arbitrary MRFs with cycles.

In this paper we provide quality guarantees for max-product fixed-points in general settings that can be calculated prior to the execution of the algorithm. To this end, we define worst-case bounds on the quality of any max-product fixed-point for any MRF, independently of its structure and parameters. Furthermore, we show how tighter guarantees can be obtained for MRFs with specific structures. For example, we prove that in 2-D grids max-product fixed points assignments have at least 33% of the quality of the optimum; and that for MRFs with large variable-disjoint cycles[1] they have at least 90% of the quality of the optimum. These results shed some light on the relationship between the quality of max-product assignments and the structure of MRFs.

Our results build upon two main components: (i) the characterization of any fixed-point max-product assignment as a neighbourhood maximum in a specific region of the MRF [17]; and (ii) the worst-case bounds on the quality of a neighbourhood maximum obtained in the K-optimality framework [18, 19]. We combine these two results by: (i) generalising the worst-case bounds in [18, 19] to consider any arbitrary region; and (ii) assessing worst-case bounds for the specific region presented in [17] (for which any fixed-point max-product assignment is known to be maximal).

## 2 Overview

### 2.1 The max-sum algorithm in Pairwise Markov Random Fields

A discrete pairwise Markov Random Field (MRF) is an undirected graphical model where each interaction is specified by a discrete potential function, defined on a single or a pair of variables. The structure of an MRF defines a graph $G = \langle V, E \rangle$, in which the nodes $V$ represent discrete variables, and edges $E$ represent interactions between nodes. Then, an MRF contains a unary potential function $\Psi_s$ for each node $s \in V$ and a pairwise potential function $\Psi_{st}$ for each edge $(s, t) \in E$; the joint probability distribution of the MRF assumes the following form:

$$p(x) = \frac{1}{Z} \prod_{s \in V} \Psi_s(x_s) \prod_{(s,t) \in E} \Psi_{st}(x_s, x_t) = \frac{1}{Z} \exp \left( \sum_{s \in V} \theta_s(x_s) + \sum_{(s,t) \in E} \theta_{st}(x_s, x_t) \right) = \frac{1}{Z} \exp \left( \theta(x) \right), \quad (1)$$

where $Z$ is a normalization constant and $\theta_s(x_s), \theta_{st}(x_s, x_t)$ stand for the logarithm of $\Psi_s(x_s), \Psi(x_s, x_t)$ which are well-defined if $\Psi_s(x_s), \Psi(x_s, x_t)$ are strictly positive.

Within this setting, the classical problem of *maximum a posteriori* (MAP) estimation corresponds to finding the most likely configuration under distribution $p(x)$ in equation 1. In more formal terms, the MAP configuration $x^* = \{x_s^* | s \in V\}$ is given by:

$$x^* \triangleq arg \max_{x \in \mathcal{X}^N} \left[ \prod_{s \in V} \Psi_s(x_s) \prod_{(s,t) \in E} \Psi_{st}(x_s, x_t) \right] \triangleq arg \max_{x \in \mathcal{X}^N} \left[ \sum_{s \in V} \theta_s(x_s) + \sum_{(s,t) \in E} \theta_{st}(x_s, x_t) \right], \quad (2)$$

where $\mathcal{X}^N$ is the Cartesian product space in which $x = \{x_s | s \in V\}$ takes values.

Note that the MAP configuration may not be unique, that is, there may be multiple configurations, that attain the maximum in equation 1. In this work we assume that: (i) there is a unique MAP assignment (as assumed in [17]); and (ii) all potentials $\theta_s$ and $\theta_{st}$ are non-negative.

The max-product algorithm is an iterative, local, message-passing algorithm for finding the MAP assignment in a discrete MRF as specified by equation 2. The max-sum algorithm is the correspondent of the max-product algorithm when we consider the log-likelihood domain. The standard update rules for max-sum algorithm are:

$$m_{ij}(x_j) = \alpha_{ij} + \max_{x_i} \left[ \theta_i(x_i) + \theta_{ij}(x_i, x_j) + \sum_{k \in N(i) \backslash j} m_{ki}(x_i) \right] \qquad b_i(x_i) = \theta_i(x_i) + \sum_{k \in N(i)} m_{ki}(x_i)$$

where $\alpha_{ij}$ is a normalization constant and $N(i)$ is the set of indices for variables that are connected to $x_i$. Here $m_{ij}(x_j)$ represents the message that variable $x_i$ sends to variable $x_j$. At the first iteration all messages are initialised to constant functions. At each following iteration, each variable $x_i$ aggregates all incoming messages and computes the belief $b_i(x_i)$, which is then used to obtain the max-sum assignment $x^{MS}$. Specifically, for every variable $x_i \in V$ we have $x_i^{MS} = arg \max_{x_i} b_i(x_i)$.

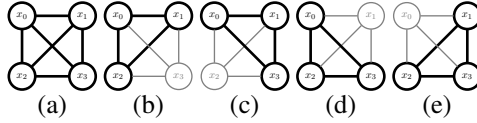

Figure 1: (a) 4-complete graph and (b)-(e) sets of variables covered by the SLT-region.

The convergence of the max-sum is usually characterized considering fixed points for the message update rules, i.e. when all the messages exchanged are equal to the last iteration. Now, the max-sum algorithm is known to be correct over acyclic and single-cycle graphs. Unfortunately, on general graphs the aggregation of messages flowing into each variable only represents an approximate solution to the maximization problem. Nonetheless, it is possible to characterise the solution obtained by max-sum as we discuss below.

## 2.2 Neighborhood maximum characterisation of max-sum fixed points

In [17], Weiss et al. characterize how well max-sum approximates the MAP assignment. In particular, they find the conditions for a fixed-point max-sum assignment $x^{MS}$ to be neighbourhood maximum, namely greater than all other assignments in a specific large region around $x^{MS}$. Notice that characterising an assignment as neighbourhood maximum is weaker than a global maximum, but stronger than a local maximum. Weiss et al. introduce the notion of *Single Loops and Trees* (SLT) region to characterise the assignments in such region.

**Definition 1** (SLT region)**.** *An SLT-region of $x$ in $\mathcal{G}$ includes all assignments $x'$ that can be obtained from $x$ by: (i) choosing an arbitrary subset $S \subseteq V$ such that its vertex-induced subgraph contains at most one cycle per connected component; (ii) assigning arbitrary values to the variables in $S$ while keeping the assignment to the other variables as in $x$.*

Hence, we say that an assignment $x^{SLT}$ is SLT-optimal if it is greater than any other assignment in its SLT region. Finally, the main result in [17] is the characterisation of any max-sum fixed-point assignments as an SLT-optimum. Figures 1(b)-(e) illustrate examples of assignments in the SLT-region in the complete graph of figure 1(a), here boldfaced nodes stand for variables that vary the assignment with respect to $x^{SLT}$.

# 3 Generalizing size and distance optimal bounds

In [18], Pearce et al. introduced worst-case bounds on the quality of a neighbourhood maximum in a region characterized by its size. Similary, Kiekintveld et al. introduced in [19] analogous worst-case bounds but using as a criterion the distance in the graph. In this section we generalize these bounds to use them for any neighbourhood maximum in a region characterized by arbitrary criteria. Concretely we show that our generalization can be used for bounding the quality of max-sum assignments.

## 3.1 $\mathcal{C}$-optimal bounds

Hereafter we propose a general notion of region optimality, the so-called $\mathcal{C}$-optimality, and describe how to calculate bounds for a $\mathcal{C}$-optimal assignment, namely an assignment that is neighbourhood maximum in a region characterized by an arbitrary $\mathcal{C}$ criteria. The concept of $\mathcal{C}$-optimality requires the introduction of several concepts.

Given $A, B \subseteq V$ we say that $B$ completely covers $A$ if $A \subseteq B$. We say that $B$ does not cover $A$ at all if $A \cap B = \emptyset$. Otherwise, we say that $B$ covers $A$ partially. A region $\mathcal{C} \subset \mathcal{P}(V)$ is a set composed by subsets of $V$. We say that $A \subseteq V$ is covered by $\mathcal{C}$ if there is a $C^{\alpha} \in \mathcal{C}$ such that $C^{\alpha}$ completely covers $A$.

Given two assignments $x^A$ and $x^B$, we define $D(x^A, x^B)$ as the set containing the variables whose values in $x^A$ and $x^B$ differ. An assignment is $\mathcal{C}$-optimal if it cannot be improved by changing the values in any group of variables covered by $\mathcal{C}$. That is, an assignment $x^A$ is $\mathcal{C}$-optimal if for every assignment $x^B$ s.t. $D(x^A, x^B)$ is covered by $\mathcal{C}$ we have that $\theta(x^A) \geq \theta(x^B)$.

For any $S \in E$ we define $cc(S, \mathcal{C}) = |\{C^{\alpha} \in \mathcal{C} \text{ s.t } S \subseteq C^{\alpha}\}|$, that is, the number of elements in $\mathcal{C}$ that cover $S$ completely. We also define $nc(S, \mathcal{C}) = |\{C^{\alpha} \in \mathcal{C} \text{ s.t } S \cap C^{\alpha} = \emptyset\}|$, that is, the number of elements in $\mathcal{C}$ that do not cover $S$ at all.

**Proposition 1.** *Let $\mathcal{G} = \langle V, E \rangle$ be a graphical model and $\mathcal{C}$ a region. If $x^{\mathcal{C}}$ is a $\mathcal{C}$-optimum then*

$$\theta(x^{\mathcal{C}}) \geq \frac{cc_*}{|\mathcal{C}| - nc_*} \theta(x^*) \tag{3}$$

*where $cc_* = \min_{S \in E} cc(S, \mathcal{C})$, $nc_* = \min_{S \in E} nc(S, \mathcal{C})$, and $x^*$ is the MAP assignment.*

*Proof.* The proof is a generalization of the one in [20] for k-optimality. For every $C^{\alpha} \in \mathcal{C}$, consider an assignment $x^{\alpha}$ such that $x_i^{\alpha} = x_i^{\mathcal{C}}$ if $x_i \notin C^{\alpha}$ and $x_i^{\alpha} = x_i^*$ if $x_i \in C^{\alpha}$. Since $x^{\mathcal{C}}$ is $\mathcal{C}$-optimal, for all $C^{\alpha} \in \mathcal{C}$, $\theta(x^{\mathcal{C}}) \geq \theta(x^{\alpha})$ holds, and hence:

$$\theta(x^{\mathcal{C}}) \geq \left( \sum_{C^{\alpha} \in \mathcal{C}} \theta(x^{\alpha}) \right) / |\mathcal{C}|. \tag{4}$$

Notice that although $\theta(x^{\alpha})$ is defined as the sum of unary potentials and pairwise potentials values we can always get rid of unary potentials by combining them into pairwise potentials without changing the structure of the MRF. In so doing, for each $x^{\alpha}$, we have that $\theta(x^{\alpha}) = \sum_{S \in E} \theta_S(x^{\alpha})$. We classify each edge $S \in E$ into one of three disjoint groups, depending on whether $C^{\alpha}$ covers $S$ completely ($T(C^{\alpha})$), partially ($P(C^{\alpha})$), or not at all ($N(C^{\alpha})$), so that $\theta(x^{\alpha}) = \sum_{S \in T(C^{\alpha})} \theta_S(x^{\alpha}) + \sum_{S \in P(C^{\alpha})} \theta_S(x^{\alpha}) + \sum_{S \in N(C^{\alpha})} \theta_S(x^{\alpha})$. We can remove the partially covered potentials at the cost of obtaining a looser bound. Hence $\theta(x^{\alpha}) \geq \sum_{S \in T(C^{\alpha})} \theta_S(x^{\alpha}) + \sum_{S \in N(C^{\alpha})} \theta_S(x^{\alpha})$. Now, by definition of $x^{\alpha}$, for every variable $x_i$ in a potential completely covered by $C^{\alpha}$ we have that $x_i^{\alpha} = x_i^*$, and for every variable $x_i$ in a potential not covered at all by $C^{\alpha}$ we have that $x_i^{\alpha} = x_i^{\mathcal{C}}$. Hence, $\theta(x^{\alpha}) \geq \sum_{S \in T(C^{\alpha})} \theta_S(x^*) + \sum_{S \in N(C^{\alpha})} \theta_S(x^{\mathcal{C}})$. To assess a bound, after substituting this inequality in equation 4, we have that:

$$\theta(x^{\mathcal{C}}) \geq \frac{\sum_{C^{\alpha} \in \mathcal{C}} \sum_{S \in T(C^{\alpha})} \theta_S(x^*) + \sum_{C^{\alpha} \in \mathcal{C}} \sum_{S \in N(C^{\alpha})} \theta_S(x^{\mathcal{C}})}{|\mathcal{C}|}. \tag{5}$$

We need to express the numerator in terms of $\theta(x^{\mathcal{C}})$ and $\theta(x^*)$. Here is where the previously defined sets $cc(S, \mathcal{C})$ and $nc(S, \mathcal{C})$ come into play. Grouping the sum by potentials and recall that $cc_* = \min_{S \in E} cc(S, \mathcal{C})$, the term on the left can be expressed as:

$$\sum_{C^{\alpha} \in \mathcal{C}} \sum_{S \in T(C^{\alpha})} \theta_S(x^*) = \sum_{S \in E} cc(S, \mathcal{C}) \cdot \theta_S(x^*) \geq \sum_{S \in E} cc_* \cdot \theta_S(x^*) = cc_* \cdot \theta(x^*).$$

Furthermore, recall that $nc_* = \min_{S \in E} nc(S, \mathcal{C})$, we can do the same with the right term:

$$\sum_{C^{\alpha} \in \mathcal{C}} \sum_{S \in N(C^{\alpha})} \theta_S(x^{\mathcal{C}}) = \sum_{S \in E} nc(S, \mathcal{C}) \cdot \theta_S(x^{\mathcal{C}}) \geq \sum_{S \in E} nc_* \cdot \theta_S(x^{\mathcal{C}}) = nc_* \cdot \theta(x^{\mathcal{C}}).$$

After substituting these two results in equation 5 and rearranging terms, we obtain equation 3. □

## 3.2 Size-optimal bounds as a specific case of $\mathcal{C}$-optimal bounds

Now we present the main result in [18] as a specific case of $\mathcal{C}$-optimality. An assignment is $k$-size-optimal if it can not be improved by changing the value of any group of size $k$ or fewer variables.

**Proposition 2.** *For any MRF and for any $k$-optimal assignment $x^k$:*

$$\theta(x^k) \geq \frac{(k-1)}{(2|V| - k - 1)} \theta(x^*) \tag{6}$$

*Proof.* This result is just a specific case of our general result where we take as a region all subsets of size $k$, that is $\mathcal{C} = \{C^{\alpha} \subseteq V \mid |C^{\alpha}| = k\}$. The number of elements in the region is $|\mathcal{C}| = \binom{|V|}{k}$. The number of elements in $\mathcal{C}$ that completely cover $S$ is $cc(S, \mathcal{C}) = \binom{|V|-2}{k-2}$ (take the two variables in $S$ plus $k-2$ variables out of the remaining $|V| - 2$). The number of elements in $\mathcal{C}$ that do not cover $S$ at all is $nc(S, \mathcal{C}) = \binom{|V|-2}{k}$ (take $k$ variables out of the remaining $|V| - 2$ variables). Finally, we obtain equation 6 by using $|V|$, $cc_*$ and $nc_*$ in equation 3, and simplifying. □

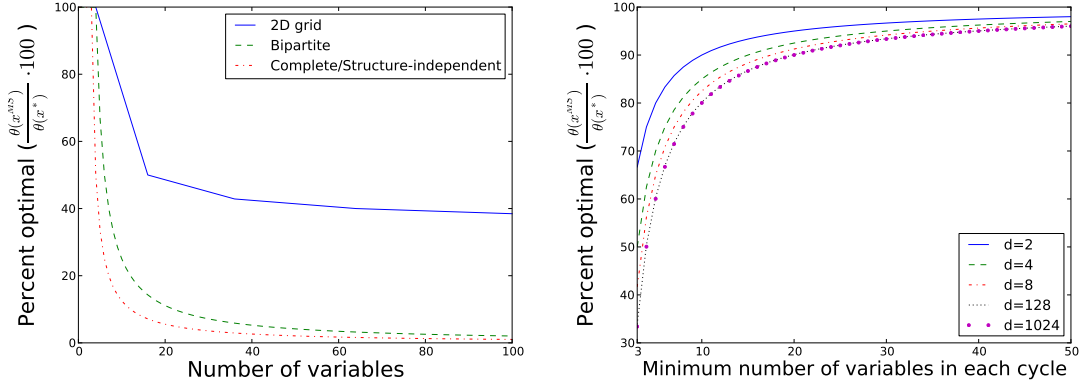

(a) Bounds on complete, bipartite and 2-D structures when varying the number of variables.

(b) Bounds on MRFs with variable-disjoint cycles when varying the number of cycles and their size.

Figure 2: Percent optimal bounds for max-sum fixed point assignments in specific MRF structures.

## 4 Quality guarantees on max-sum fixed-point assignments

In this section we define quality guarantees for max-sum fixed-point assignments in MRFs with arbitary and specific structures. Our quality guarantees prove that the value of any max-sum fixed-point assignments can not be less than a fraction of the optimum.

The main idea is that by virtue of the characterization of any max-sum fixed point assignment as SLT-optimal, we can select any region $\mathcal{C}$ composed of a combination of single cycles and trees of our graph and use it for computing its corresponding $\mathcal{C}$-optimal bound by means of proposition 1.

We start by proving that bounds for a given graph apply to its subgraphs. Then, we find that the bound for the complete graph applies to any MRF independently of its structure and parameters. Afterwards we provide tighter bounds for MRFs with specific structures.

### 4.1 $\mathcal{C}$-optimal bounds based on the SLT region

In this section we show that $\mathcal{C}$-optimal bounds based on SLT-optimality for a given graph can be applied to any of its subgraphs.

**Proposition 3.** *Let $\mathcal{G} = \langle V, E \rangle$ be a graphical model and $\mathcal{C}$ the SLT-region of $\mathcal{G}$. Let $\mathcal{G}' = \langle V', E' \rangle$ be a subgraph of $\mathcal{G}$. Then the bound of equation 3 for $\mathcal{G}$ holds for any SLT-optimal assignment in $\mathcal{G}'$.*

*Sketch of the proof.* We can compose a region $\mathcal{C}'$ containing the same elements as $\mathcal{C}$ but removing those variables which are not contained in $V'$. Note that SLT-optimality on $\mathcal{G}'$ guarantees optimality in each element of $\mathcal{C}'$. Observe that the bound obtained by applying equation 3 to $\mathcal{C}'$ is greater or equal than the bound obtained for $\mathcal{C}$. Hence, the bound for $\mathcal{G}$ applies also to $\mathcal{G}'$.

A direct conclusion of proposition 3 is that any bound based on the SLT-region of a complete graph of $n$ variables can be directly applied to any subgraph of $n$ or fewer variables regardless of its structure. In what follows we assess the bound for a complete graph.

**Proposition 4.** *Let $\mathcal{G} = \langle V, E \rangle$ be a complete MRF. For any max-sum fixed point assignment $x^{MS}$,*

$$\theta(x^{MS}) \geq \frac{1}{|V| - 2} \cdot \theta(x^*). \tag{7}$$

*Proof.* Let $\mathcal{C}$ be a region containing every possible combination of three variables in $V$. Every set of three variables is part of the SLT-region because it can contain at most one cycle. The development in the proof of proposition 2 can be applied here for $k = 3$ to obtain equation 7. $\square$

**Corollary 5.** *For any MRF, any max-sum fixed point assignment $x^{MS}$ satisfies equation 7.*

Since any graph can be seen as a subgraph of the complete graph with the same number of variables, the corollary is straightforward given propositions 3 and 4. Figure 2(a) plots this structure-independent bound when varying the number of variables. Observe that it rapidly decreases with

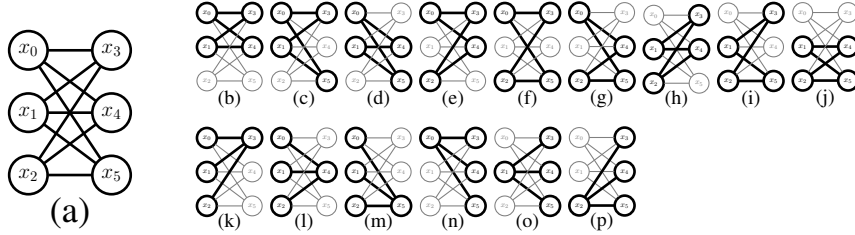

Figure 3: Example of (a) a 3-3 bipartite graph and (b)-(p) sets of variables covered by the SLT-region.

the number of variables and it is only significant on very small MRFs. In the next section, we show how to exploit the knowledge of the structure of an MRF to improve the bound's significance.

### 4.2 SLT-bounds for specific MRF structures and independent of the MRF parameters

In this section we show that for MRFs with specific structures, it is possible to provide bounds much tighter than the structure-independent bound provided by corollary 5. These structures include, but are not limited to, bipartite graphs, 2-D grids, and variable-disjoint cycle graphs.

#### 4.2.1 Bipartite graphs

In this section we define the $\mathcal{C}$-optimal bound of equation 3 for any max-sum fixed point assignment in an $n$-$m$ bipartite MRF. An $n$-$m$ bipartite MRF is a graph whose vertices can be divided into two disjoint sets, one with $n$ variables and another one with $m$ variables, such that the $n$ variables in the first set are connected to the $m$ variables in the second set. Figure 3(a) depicts a 3-3 bipartite MRF.

**Proposition 6.** *For any MRF with $n$-$m$ bipartite structure where $m \geq n$, and for any max-sum fixed point assignment $x^{MS}$ we have that:*

$$\theta(x^{MS}) \geq b(n,m) \cdot \theta(x^*) \qquad b(n,m) = \begin{cases} \frac{1}{n} & m \geq n+3 \\ \frac{2}{n+m-2} & m < n+3 \end{cases} \qquad (8)$$

*Proof.* Let $\mathcal{C}^A$ be a region including one out of the $n$ variables and all of the $m$ variables (in figure 3, elements (n)-(p)). Since the elements of this region are trees, we can guarantee optimality on them. The number of elements of the region is $|\mathcal{C}^A| = n$. It is clear that each edge in the graph is completely covered by one of the elements of $\mathcal{C}^A$, and hence $cc_* = 1$. Furthermore, every edge is partially covered, since all of the $m$ variables are present in every element, and hence $nc_* = 0$. Applying equation 3 gives the bound $1/n$.

Alternatively, we can define a region $\mathcal{C}^B$ formed by taking sets of four variables, two from each set. Since the elements of $\mathcal{C}^B$ are single-cycle graphs (in figure 3, elements (b)-(j)), we can guarantee optimality on them. Applying proposition 1, we obtain the bound $\frac{2}{n+m-2}$. Observe that $\frac{2}{n+m-2} > \frac{1}{n}$ when $m < n+3$, and so equation 8 holds (details can be found in the additional material). □

**Example 1.** *Consider the 3-3 bipartite MRF of figure 3(a). Figures 3(b)-(j) show the elements in the region $\mathcal{C}^B$ composed of sets of four variables, two from each side. Therefore $|\mathcal{C}^B|$ is 9. Then, for any edge $S \in E$ there are 4 sets in $\mathcal{C}^B$ that contain its two variables. For example, the edge that links the upper left variable ($x_0$) and the upper right variable ($x_3$) is included in the subgraphs of figures 3(b), (c), (e) and (f). Moreover, for any edge $S \in E$ there is a single element in $\mathcal{C}^B$ that does not cover it at all. For example, the only graph that does not include neither $x_0$ nor $x_3$ is the graph of figure 3(j). Thus, the bound is $4/(9-1) = 1/2$.*

Figure 2(a) plots the bound of equation 8 for bipartite graphs when varying the number of variables. Note that although, also in this case, the value of the bound rapidly decreases with the number of variables, it is two times the values of the structure-independent bound (see equation 7).

#### 4.2.2 Two-dimensional (2-D) grids

In this section we define the $\mathcal{C}$-optimal bound of equation 3 for any max-sum fixed point assignment in a two-dimensional grid MRF. An $n$-grid structure stands for a graph with $n$ rows and $n$ columns where each variable has 4 neighbours. Figure 4 (a) depicts a 4-grid MRF.

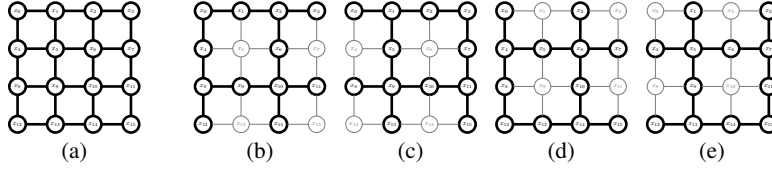

Figure 4: Example of (a) a 4-grid graph and (b)-(e) sets of variables covered by the SLT-region.

**Proposition 7.** *For any MRF with an $n$ grid structure where $n$ is an even number, for any max-sum fixed point assignment $x^{MS}$ we have that*

$$\theta(x^{MS}) \geq \frac{n}{3n-4} \cdot \theta(x^*) \qquad (9)$$

*Proof.* We can partition columns in pairs joining column 1 with column $(n/2)+1$, column 2 with column $(n/2)+2$ and so on. We can partition rows in the same way. Let $\mathcal{C}$ be a region where each element contains the vertices in a pair of rows at distance $\frac{n}{2}$ together with those in a pair of columns at distance $\frac{n}{2}$. Note that optimality is guaranteed in each $C^\alpha \in \mathcal{C}$ because variables in two non-consecutive rows and two non-consecutive columns create a single-cycle graph. Since we take every possible combination, $|\mathcal{C}| = (\frac{n}{2})^2$. Each edge is completely covered by $\frac{n}{2}$ elements and hence $cc_* = \frac{n}{2}$. Finally[2], for each edge $S$, there are $nc_* = (\frac{n}{2}-1)(\frac{n}{2}-2)$ elements of $\mathcal{C}$ that do not cover $S$ at all. Substituting these values into equation 3 leads to equation 9. $\square$

**Example 2.** *Consider the 4-grid MRF of figure 4 (a). Figures 4 (b)-(e) show the vertex-induced subgraphs for each set of vertices in the region $\mathcal{C}$ formed by the combination of any pairs of rows in $\{(1,3),(2,4)\}$ and pair of columns in $\{(1,3),(2,4)\}$. Therefore $|\mathcal{C}| = 4$. Then, for any edge $S \in E$ there are 2 sets that contain its two variables. For example, the edge that links the two first variables in the first row, namely $x_0$ and $x_1$, is included in the subgraphs of figures (a) and (b). Moreover, for any edge $S \in E$ there is no set that contains no variable from $S$. Thus, the bound is $1/2$.*

Figure 2(a) plots the bound for 2-D grids when varying the number of variables. Note that when compared with the bound for complete and bipartite structures, the bound for 2-D grids decreases smoothly and tends to stabilize as the number of variables increases. In fact, observe that by equation 9, the bound for 2-D grids is never less that $1/3$ independently of the grid size.

### 4.2.3 MRFs that are a union of variable-disjoint cycles

In this section we assess a bound for MRFs composed of a set of variable-disjoint cycles, namely of cycles that do not share any variable.

A common pattern shared by the bounds assessed so far is that they decrease as the number of variables of an MRF grows. This section provides an example showing that there are specific structures for which $\mathcal{C}$-optimality obtains significant bounds for large MRFs.

**Example 3.** *Consider the MRF composed of two variable-disjoint cycles of size 4 depicted in figure 5(a). To create the region, we remove each of the variables of the first cycle, one at a time (see figures 5(b)-(e)). We act analogously with the second cycle. Hence, $\mathcal{C}$ is composed of 8 elements. Just by counting we observe that each edge is completely covered 6 times, so $cc_* = 6$. Since we are removing a single variable at a time, $nc_* = 0$. Hence, the bound for a max-sum fixed point in this MRF structure is $6/8 = 3/4$.*

The following result generalizes the previous example to MRFs containing $d$ variable-disjoint cycles of size larger or equal to $l$.

**Proposition 8.** *For any MRF such that every pair of cycles is variable-disjoint and where there are at most $d$ cycles of size $l$ or larger, and for any max-sum fixed point assignment $x^{MS}$, we have that:*

$$\theta(x^{MS}) \geq \left(1 - \frac{2(d-1)}{d \cdot l}\right) \cdot \theta(x^*) = \frac{(l-2) \cdot d + 2}{l \cdot d} \cdot \theta(x^*). \qquad (10)$$

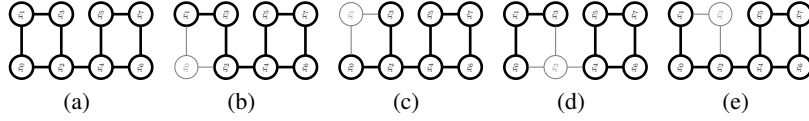

Figure 5: (a) 2 variable-disjoint cycles MRF of size 4 and (b-e) sets of variables covered by the SLT-region.

The proof generalizes the region explained in example 3 to any variable-disjoint cycle MRF by defining a region that includes an element for every possible edge removal from every cycle but one. The proof is omitted here due to lack of space but can be consulted in the additional material.

Equation 10 shows that the bound: (i) decreases with the number of cycles; and (ii) increases as the maximum number of variables in each cycle grows. Figure 2(b) illustrates the relationship between the bound, the number of cycles ($d$), and the maximum size of the cycles ($l$). The first thing we observe is that the size of the cycles has more impact on the bound than the number of cycles. In fact, observe that by equation 10, the bound for a variable-disjoint cycle graph with a maximum cycle size of $l$ is at least $\frac{(l-2)}{l}$, independently of the number of cycles. Thus, if the minimum size of a cycle is 20, the quality for a fixed point is guaranteed to be at least $90\%$. Hence, quality guarantees for max-sum fixed points are good whenever: (i) the cycles in the MRF do not share any variables; and (ii) the smallest cycle in the MRF is large. Therefore, our result confirms and refines the recent results obtained for single-cycle MRFs [11].

### 4.3 SLT-bounds for arbitrary MRF structures and independent of the MRF parameters

In this section we discuss how to assess tight SLT-bounds for any arbitrary MRF structure. Similarly to [18, 20], we can use linear fractional programming (LFP) to compute the structure specific SLT bounds in any MRF with arbitrary structure. Let $\mathcal{C}$ be a region for all subsets in the SLT region of the graphical model $\mathcal{G} = \langle V, E \rangle$ of an MRF. For each $S \in E$, the LFP contains two LFP variables that represents the value of the edge $S$ for the SLT-optimum, $x^{MS}$, and for the MAP assignment, $x^*$. The objective of the LFP is to minimize $\frac{\sum_{S \in E} \theta_S(x^{MS})}{\sum_{S \in E} \theta_S(x^*)}$ such that for all $\mathcal{C}^\alpha \in \mathcal{C}$, $\theta(x^{MS}) - \theta(x^\alpha) \geq 0$. Following [18, 20], for each $\mathcal{C}^\alpha \in \mathcal{C}$, $\theta(x^\alpha)$ can be expressed in terms of the value of the potentials for $x^{MS}$ and $x^*$. Then, the optimal value of this LFP is a tight bound for any MRF with the given specific structure. Indeed, the solution of the LFP provides the values of potentials for $x^{MS}$ and $x^*$ that produce the worst-case MRF whose SLT-optimum has the lowest value with respect to the optimum. However, because this method requires to list all the sets in the SLT-region, the complexity of generating an LFP increases exponentially with the number of variables in the MRF. Therefore, although this method provides more flexibility to deal with any arbitrary structure, its computational cost does not scale with the size of MRFs in contrast with the structure specific SLT-bounds of section 4.2, that are assessed in constant time.

## 5 Conclusions

We provided worst-case bounds on the quality of any max-product fixed point. With this aim, we have introduced $\mathcal{C}$-optimality, which has proven a valuable tool to bound the quality of max-product fixed points. Concretely, we have proven that independently of an MRF structure, max-product has a quality guarantee that decreases with the number of variables of the MRF. Furthermore, our results allow to identify new classes of MRF structures, besides acyclic and single-cycle, for which we can provide theoretical guarantees on the quality of max-product assignments. As an example, we defined significant bounds for 2-D grids and MRFs with variable-disjoint cycles.

**Acknowledgments**

Work funded by projects EVE (TIN2009-14702-C02-01,TIN2009-14702-C02-02), AT(CONSOLIDER CSD2007-0022), and Generalitat de Catalunya (2009-SGR-1434). Vinyals is supported by the Ministry of Education of Spain (FPU grant AP2006-04636).

## Footnotes

[1]MRFs in which all cycles are variable-disjoint, namely that they do not share any edge and in which each cycle contains at least 20 variables.

[2]Details can be found in the additional material

# References

[1] Marshall F. Tappen and William T. Freeman. Comparison of graph cuts with belief propagation for stereo, using identical mrf parameters. In *In ICCV*, pages 900–907, 2003.

[2] Jon Feldman, Martin J. Wainwright, and David R. Karger. Using linear programming to decode binary linear codes. *IEEE Transactions on Information Theory*, 51(3):954–972, 2005.

[3] Chen Yanover and Yair Weiss. Approximate inference and protein-folding. In *Advances in Neural Information Processing Systems*, pages 84–86. MIT Press, 2002.

[4] Alessandro Farinelli, Alex Rogers, Adrian Petcu, and Nicholas R. Jennings. Decentralised coordination of low-power embedded devices using the max-sum algorithm. In *AAMAS*, pages 639–646, 2008.

[5] Solomon Eyal Shimony. Finding MAPs for belief networks is NP-Hard. *Artif. Intell.*, 68(2):399–410, 1994.

[6] Judea Pearl. *Probabilistic Reasoning in Intelligent Systems*. Morgan Kaufmann Publishers Inc., San Francisco, CA, USA, 1988.

[7] Srinivas M. Aji and Robert J. McEliece. The generalized distributive law. *IEEE Transactions on Information Theory*, 46(2):325–343, 2000.

[8] Srinivas Aji, Gavin Horn, Robert Mceliece, and Meina Xu. Iterative min-sum decoding of tail-biting codes. In *In Proc. IEEE Information Theory Workshop*, pages 68–69, 1998.

[9] Brendan J. Frey, Ralf Koetter, G. David Forney Jr., Frank R. Kschischang, Robert J. McEliece, and Daniel A. Spielman. Introduction to the special issue on codes on graphs and iterative algorithms. *IEEE Transactions on Information Theory*, 47(2):493–497, 2001.

[10] Brendan J. Frey, Ralf Koetter, and Nemanja Petrovic. Very loopy belief propagation for unwrapping phase images. In *NIPS*, pages 737–743, 2001.

[11] Yair Weiss. Correctness of local probability propagation in graphical models with loops. *Neural Computation*, 12(1):1–41, 2000.

[12] Mohsen Bayati, Christian Borgs, Jennifer T. Chayes, and Riccardo Zecchina. Belief-propagation for weighted b-matchings on arbitrary graphs and its relation to linear programs with integer solutions. *CoRR*, abs/0709.1190, 2007.

[13] Sujay Sanghavi, Dmitry Malioutov, and Alan Willsky. Linear programming analysis of loopy belief propagation for weighted matching. In J.C. Platt, D. Koller, Y. Singer, and S. Roweis, editors, *Advances in Neural Information Processing Systems 20*, pages 1273–1280. MIT Press, Cambridge, MA, 2008.

[14] Sujay Sanghavi, Devavrat Shah, and Alan S. Willsky. Message-passing for maximum weight independent set. *CoRR*, abs/0807.5091, 2008.

[15] Bert Huang and Tony Jebara. Loopy belief propagation for bipartite maximum weight b-matching. In Marina Meila and Xiaotong Shen, editors, *In Proceedings of the Eleventh International Conference on Artificial Intelligence and Statistics*, March 2007.

[16] Martin J. Wainwright, Tommi Jaakkola, and Alan S. Willsky. Tree consistency and bounds on the performance of the max-product algorithm and its generalizations. *Statistics and Computing*, 14(2):143–166, 2004.

[17] Yair Weiss and William T. Freeman. On the optimality of solutions of the max-product belief-propagation algorithm in arbitrary graphs. *IEEE Transactions on Information Theory*, 47(2):736–744, 2001.

[18] Jonathan P. Pearce and Milind Tambe. Quality guarantees on k-optimal solutions for distributed constraint optimization problems. In *IJCAI*, pages 1446–1451, 2007.

[19] Christopher Kiekintveld, Zhengyu Yin, Atul Kumar, and Milind Tambe. Asynchronous algorithms for approximate distributed constraint optimization with quality bounds. In *AAMAS*, pages 133–140, 2010.

[20] J. P. Pearce. *Local Optimization in Cooperative Agent Networks*. PhD thesis, University of Southern California, Los Angeles, CA, August 2007.

